# $H\infty$ Optimality Criteria for LMS and Backpropagation

**Babak Hassibi**
Information Systems Laboratory
Stanford University
Stanford, CA 94305

**Ali H. Sayed**
Dept. of Elec. and Comp. Engr.
University of California Santa Barbara
Santa Barbara, CA 93106

**Thomas Kailath**
Information Systems Laboratory
Stanford University
Stanford, CA 94305

## Abstract

We have recently shown that the widely known LMS algorithm is an $H^\infty$ optimal estimator. The $H^\infty$ criterion has been introduced, initially in the control theory literature, as a means to ensure robust performance in the face of model uncertainties and lack of statistical information on the exogenous signals. We extend here our analysis to the nonlinear setting often encountered in neural networks, and show that the backpropagation algorithm is *locally* $H^\infty$ optimal. This fact provides a theoretical justification of the widely observed excellent robustness properties of the LMS and backpropagation algorithms. We further discuss some implications of these results.

## 1 Introduction

The LMS algorithm was originally conceived as an approximate recursive procedure that solves the following problem (Widrow and Hoff, 1960): given a sequence of $n \times 1$ input column vectors $\{h_i\}$, and a corresponding sequence of desired scalar responses $\{d_i\}$, find an estimate of an $n \times 1$ column vector of weights $w$ such that the sum of squared errors, $\sum_{i=0}^{N}|d_i - h_i^T w|^2$, is minimized. The LMS solution recursively

updates estimates of the weight vector along the direction of the instantaneous gradient of the squared error. It has long been known that LMS is an approximate minimizing solution to the above least-squares (or $H^2$) minimization problem. Likewise, the celebrated backpropagation algorithm (Rumelhart and McClelland, 1986) is an extension of the gradient-type approach to nonlinear cost functions of the form $\sum_{i=0}^{N} |d_i - h_i(w)|^2$, where $h_i(.)$ are known nonlinear functions (*e.g.*, sigmoids). It also updates the weight vector estimates along the direction of the instantaneous gradients.

We have recently shown (Hassibi, Sayed and Kailath, 1993a) that the LMS algorithm is an $H^\infty$−optimal filter, where the $H^\infty$ norm has recently been introduced as a robust criterion for problems in estimation and control (Zames, 1981). In general terms, this means that the LMS algorithm, which has long been regarded as an approximate least-mean squares solution, is in fact a minimizer of the $H^\infty$ error norm and not of the $H^2$ norm. This statement will be made more precise in the next few sections. In this paper, we extend our results to a nonlinear setting that often arises in the study of neural networks, and show that the backpropagation algorithm is a locally $H^\infty$-optimal filter. These facts readily provide a theoretical justification for the widely observed excellent robustness and tracking properties of the LMS and backpropagation algorithms, as compared to, for example, exact least squares methods such as RLS (Haykin, 1991).

In this paper we attempt to introduce the main concepts, motivate the results, and discuss the various implications. We shall, however, omit the proofs for reasons of space. The reader is refered to (Hassibi et al. 1993a), and the expanded version of this paper for the necessary details.

## 2  Linear $H^\infty$ Adaptive Filtering

We shall begin with the definition of the $H^\infty$ norm of a transfer operator. As will presently become apparent, the motivation for introducing the $H^\infty$ norm is to capture the worst case behaviour of a system.

Let $h_2$ denote the vector space of square-summable complex-valued causal sequences $\{f_k, 0 \le k < \infty\}$, viz.,

$$h_2 = \{\text{set of sequences } \{f_k\} \text{ such that } \sum_{k=0}^{\infty} f_k^* f_k < \infty\}$$

with inner product $< \{f_k\}, \{g_k\} > = \sum_{k=0}^{\infty} f_k^* g_k$, where $*$ denotes complex conjugation. Let $T$ be a transfer operator that maps an input sequence $\{u_i\}$ to an output sequence $\{y_i\}$. Then the $H^\infty$ norm of $T$ is equal to

$$\|T\|_\infty = \sup_{u \ne 0, u \in h_2} \frac{\|y\|_2}{\|u\|_2}$$

where the notation $\|u\|_2$ denotes the $h_2$−norm of the causal sequence $\{u_k\}$, viz.,

$$\|u\|_2^2 = \sum_{k=0}^{\infty} u_k^* u_k$$

The $H^\infty$ norm may thus be regarded as the maximum *energy gain* from the input $u$ to the output $y$.

Suppose we observe an output sequence $\{d_i\}$ that obeys the following model:

$$d_i = h_i^T w + v_i \tag{1}$$

where $h_i^T = [\ h_{i1} \quad h_{i2} \quad \ldots \quad h_{in}\ ]$ is a known input vector, $w$ is an unknown weight vector, and $\{v_i\}$ is an unknown disturbance, which may also include modeling errors. We shall not make any assumptions on the noise sequence $\{v_i\}$ (such as whiteness, normally distributed, etc.).

Let $w_i = \mathcal{F}(d_0, d_i, \ldots, d_i)$ denote the estimate of the weight vector $w$ given the observations $\{d_j\}$ from time 0 up to and including time $i$. The objective is to determine the functional $\mathcal{F}$, and consequently the estimate $w_i$, so as to minimize a certain norm defined in terms of the prediction error

$$e_i = h_i^T w - h_i^T w_{i-1}$$

which is the difference between the true (uncorrupted) output $h_i^T w$ and the predicted output $h_i^T w_{i-1}$. Let $T$ denote the transfer operator that maps the unknowns $\{w - w_{-1}, \{v_i\}\}$ (where $w_{-1}$ denotes an initial guess of $w$) to the prediction errors $\{e_i\}$. The $H^\infty$ estimation problem can now be formulated as follows.

**Problem 1 (Optimal $H^\infty$ Adaptive Problem)** *Find an $H^\infty$-optimal estimation strategy $w_i = \mathcal{F}(d_0, d_1, \ldots, d_i)$ that minimizes $\|T\|_\infty$, and obtain the resulting*

$$\gamma_0^2 = \inf_{\mathcal{F}} \ \|T\|_\infty^2 = \inf_{\mathcal{F}} \ \sup_{w,v \in h_2} \ \frac{\|e\|_2^2}{\mu^{-1}|w - w_{-1}|^2 + \|v\|_2^2} \tag{2}$$

*where $|w - w_{-1}|^2 = (w - w_{-1})^T (w - w_{-1})$, and $\mu$ is a positive constant that reflects apriori knowledge as to how close $w$ is to the initial guess $w_{-1}$.*

Note that the infimum in (2) is taken over all *causal* estimators $\mathcal{F}$. The above problem formulation shows that $H^\infty$ optimal estimators guarantee the smallest prediction error energy over *all* possible disturbances of fixed energy. $H^\infty$ estimators are thus over conservative, which reflects in a more robust behaviour to disturbance variation.

Before stating our first result we shall define the input vectors $\{h_i\}$ *exciting* if, and only if,

$$\lim_{N \to \infty} \sum_{i=0}^{N} h_i^T h_i = \infty$$

**Theorem 1 (LMS Algorithm)** *Consider the model (1), and suppose we wish to minimize the $H^\infty$ norm of the transfer operator from the unknowns $w - w_{-1}$ and $v_i$ to the prediction errors $e_i$. If the input vectors $h_i$ are exciting and*

$$0 < \mu < \inf_i \frac{1}{h_i^T h_i} \tag{3}$$

*then the minimum $H^\infty$ norm is $\gamma_{opt} = 1$. In this case an optimal $H^\infty$ estimator is given by the LMS algorithm with learning rate $\mu$, viz.*

$$w_i = w_{i-1} + \mu h_i(d_i - h_i^T w_{i-1}) \quad , \quad w_{-1} \tag{4}$$

In other words, the result states that the LMS algorithm is an $H^\infty$—optimal filter. Moreover, Theorem 1 also gives an upper bound on the learning rate $\mu$ that ensures the $H^\infty$ optimality of LMS. This is in accordance with the well-known fact that LMS behaves poorly if the learning rate is too large.

Intuitively it is not hard to convince oneself that $\gamma_{opt}$ cannot be less than one. To this end suppose that the estimator has chosen some initial guess $w_{-1}$. Then one may conceive of a disturbance that yields an observation that coincides with the output expected from $w_{-1}$, i.e.

$$h_i^T w_{-1} = h_i^T w + v_i = d_i$$

In this case one expects that the estimator will not change its estimate of $w$, so that $w_i = w_{-1}$ for all $i$. Thus the prediction error is

$$e_i = h_i^T w - h_i^T w_{i-1} = h_i^T w - h_i^T w_{-1} = -v_i$$

and the ratio in (2) can be made arbitrarily close to one.

The surprising fact though is that $\gamma_{opt}$ is one and that the LMS algorithm achieves it. What this means is that LMS guarantees that the energy of the prediction error will never exceed the energy of the disturbances. This is not true for other estimators. For example, in the case of the recursive least-squares (RLS) algorithm, one can come up with a disturbance of arbitrarily small energy that will yield a prediction error of large energy.

To demonstrate this, we consider a special case of model (1) where $h_i$ is now a scalar that randomly takes on the values $+1$ or $-1$. For this model $\mu$ must be less than 1 and we chose the value $\mu = .9$. We compute the $H^\infty$ norm of the transfer operator from the disturbances to the prediction errors for both RLS and LMS. We also compute the worst case RLS disturbance, and show the resulting prediction errors. The results are illustrated in Fig. 1. As can be seen, the $H^\infty$ norm in the RLS case increases with the number of observations, whereas in the LMS case it remains constant at one. Using the worst case RLS disturbance, the prediction error due to the LMS algorithm goes to zero, whereas the prediction error due to the RLS algorithm does not. The form of the worst case RLS disturbance is also interesting; it competes with the true output early on, and then goes to zero.

We should mention that the LMS algorithm is only one of a family of $H^\infty$ optimal estimators. However, LMS corresponds to what is called the *central* solution, and has the additional properties of being the maximum entropy solution and the risk-sensitive optimal solution (Whittle 1990, Glover and Mustafa 1989, Hassibi et al. 1993b).

If there is no disturbance in (1) we have the following

**Corollary 1** *If in addition to the assumptions of Theorem 1 there is no disturbance in (1), then LMS guarantees* $\| e \|_2^2 \leq \mu^{-1} |w - w_{-1}|^2$, *meaning that the prediction error converges to zero.*

Note that the above Corollary suggests that the larger $\mu$ is (provided (3) is satisfied) the faster the convergence will be.

Before closing this section we should mention that if instead of the prediction error one were to consider the filtered error $e_{f,i} = h_i w - h_i w_i$, then the $H^\infty$ optimal estimator is the so-called normalized LMS algorithm (Hassibi et al. 1993a).

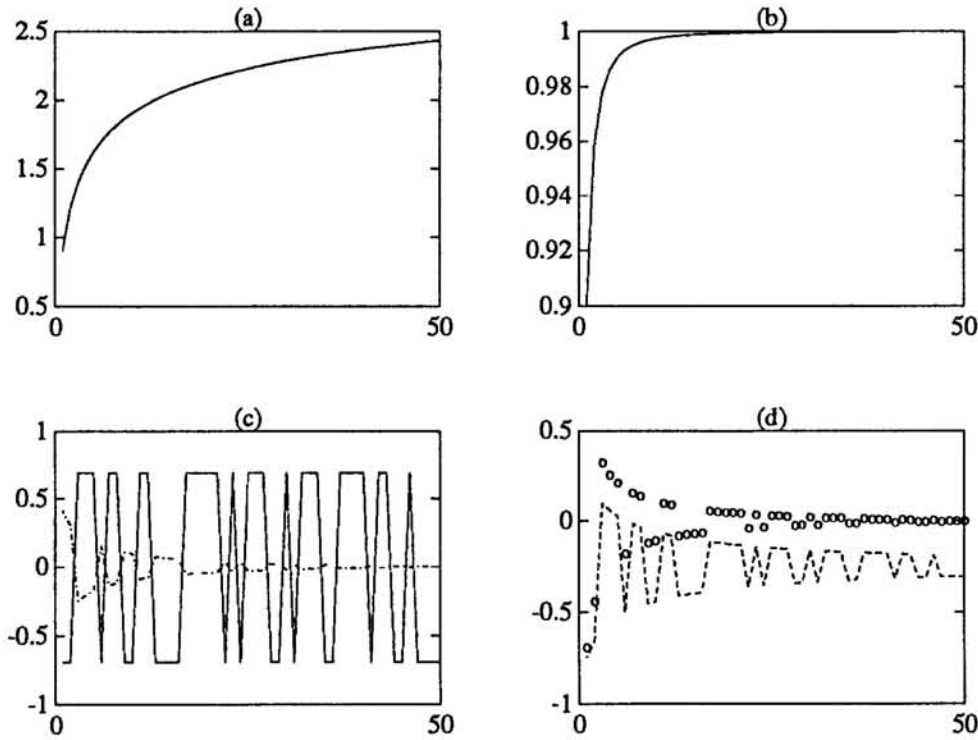

Figure 1: $H^\infty$ norm of transfer operator as a function of the number of observations for (a) RLS, and (b) LMS. The true output and the worst case disturbance signal (dotted curve) for RLS are given in (c). The predicted errors for the RLS (dashed) and LMS (dotted) algorithms corresponding to this disturbance are given in (d). The LMS predicted error goes to zero while the RLS predicted error does not.

## 3   Nonlinear $H^\infty$ Adaptive Filtering

In this section we suppose that the observed sequence $\{d_i\}$ obeys the following nonlinear model

$$d_i = h_i(w) + v_i \tag{5}$$

where $h_i(.)$ is a known *nonlinear* function (with bounded first and second order derivatives), $w$ is an unknown weight vector, and $\{v_i\}$ is an unknown disturbance sequence that includes noise and/or modelling errors. In a neural network context the index $i$ in $h_i(.)$ will correspond to the nonlinear function that maps the weight vector to the output when the $i$th input pattern is presented, i.e., $h_i(w) = h(x^{(i)}, w)$ where $x^{(i)}$ is the $i$th input pattern. As before we shall denote by $w_i = \mathcal{F}(d_0, \ldots, d_i)$ the estimate of the weight vector using measurements up to and including time $i$, and the prediction error by

$$e_i' = h_i(w) - h_i(w_{i-1})$$

Let $T$ denote the transfer operator that maps the unknowns/disurbances $\{w - w_{-1}, \{v_i\}\}$ to the prediction errors $\{e_i'\}$.

**Problem 2 (Optimal Nonlinear $H^\infty$ Adaptive Problem)** *Find an $H^\infty$-optimal estimation strategy $w_i = \mathcal{F}(d_0, d_1, \ldots, d_i)$ that minimizes $\|T\|_\infty$,*

*and obtain the resulting*

$$\gamma_0^2 = \inf_{\mathcal{F}} \ \|T\|_\infty^2 = \inf_{\mathcal{F}} \ \sup_{w,v \in h_2} \ \frac{\|e'\|_2^2}{\mu^{-1}|w - w_{-1}|^2 + \|v\|_2^2} \tag{6}$$

Currently there is no general solution to the above problem, and the class of non-linear functions $h_i(.)$ for which the above problem has a solution is not known (Ball and Helton, 1992).

To make some headway, though, note that by using the mean value theorem (5) may be rewritten as

$$d_i = h_i(w_{i-1}) + \frac{\partial h_i}{\partial w}^T (w_{i-1}^*).(w - w_{i-1}) + v_i \tag{7}$$

where $w_{i-1}^*$ is a point on the line connecting $w$ and $w_{i-1}$. Theorem 1 applied to (7) shows that the recursion

$$w_i = w_{i-1} + \mu \frac{\partial h_i}{\partial w}(w_{i-1}^*)(d_i - h_i(w_{i-1})) \tag{8}$$

will yield $\gamma = 1$. The problem with the above algorithm is that the $w_i^*$'s are not known. But it suggests that the $\gamma_{opt}$ in Problem 2 (if it exists) cannot be less than one. Moreover, it can be seen that the backpropagation algorithm is an approximation to (8) where $w_i^*$ is replaced by $w_i$. To pursue this point further we use again the mean value theorem to write (5) in the alternative form

$$d_i = h_i(w_{i-1}) + \frac{\partial h_i}{\partial w}^T (w_{i-1}).(w - w_{i-1}) + \frac{1}{2}(w - w_{i-1})^T . \frac{\partial^2 h_i}{\partial w^2}(\bar{w}_{i-1}).(w - w_{i-1}) + v_i \tag{9}$$

where once more $\bar{w}_{i-1}$ lies on the line connecting $w_{i-1}$ and $w$. Using (9) and Theorem 1 we have the following result.

**Theorem 2 (Backpropagation Algorithm)** *Consider the model (5) and the backpropagation algorithm*

$$w_i = w_{i-1} + \mu \frac{\partial h_i}{\partial w}(w_{i-1})(d_i - h_i(w_{i-1})) \quad , \quad w_{-1} \tag{10}$$

*then if the $\frac{\partial h_i}{\partial w}(w_{i-1})$ are exciting, and*

$$0 < \mu < \inf_i \frac{1}{\frac{\partial h_i}{\partial w}^T (w_{i-1}).\frac{\partial h_i}{\partial w}(w_{i-1})} \tag{11}$$

*then for all nonzero $w, v \in h_2$:*

$$\frac{\| \frac{\partial h_i}{\partial w}^T (w_{i-1})(w - w_{i-1}) \|_2^2}{\mu^{-1}|w - w_{-1}|^2 + \| v_i + \frac{1}{2}(w - w_{i-1})^T \frac{\partial^2 h_i}{\partial w^2}(\bar{w}_{i-1}).(w - w_{i-1}) \|_2^2} \leq 1$$

*where*

$$(w - w_{i-1})^T . \frac{\partial^2 h_i}{\partial w^2}(\bar{w}_{i-1}).(w - w_{i-1}) = h_i(w) - h_i(w_{i-1}) - \frac{\partial h_i}{\partial w}^T (w_{i-1}).(w - w_{i-1})$$

The above result means that if one considers a new disturbance $v_i' = v_i + \frac{1}{2}(w - w_{i-1})^T \frac{\partial^2 h_i}{\partial w^2}(\bar{w}_{i-1}).(w - w_{i-1})$, whose second term indicates how far $h_i(w)$ is from a *first order approximation* at point $w_{i-1}$, then backpropagation guarantees that the energy of the linearized prediction error $\frac{\partial h_i}{\partial w}^T(w_{i-1})(w - w_{i-1})$ does not exceed the energy of the new disturbances $w - w_{-1}$ and $v_i'$.

It seems plausible that if $w_{-1}$ is close enough to $w$ then the second term in $v_i'$ should be small and the true and linearized prediction errors should be close, so that we should be able to bound the ratio in (6). Thus the following result is expected, where we have defined the vectors $\{h_i\}$ *persistently exciting* if, and only if, for all $a \in \mathcal{R}^n$

$$\lim_{N \to \infty} a^T \left[ \sum_{i=0}^{\infty} h_i h_i^T \right] a = \infty.$$

**Theorem 3 (Local $H^\infty$ Optimality)** *Consider the model (5) and the backpropagation algorithm (10). Suppose that the $\frac{\partial h_i}{\partial w}(w_{i-1})$ are persistently exciting, and that (11) is satisfied. Then for each $\epsilon > 0$, there exist $\delta_1, \delta_2 > 0$ such that for all $|w - w_{-1}| < \delta_1$ and all $v \in h_2$ with $|v_i| < \delta_2$, we have*

$$\frac{\| e_i' \|_2^2}{\mu^{-1}|w - w_{-1}|^2 + \| v \|_2^2} \leq 1 + \epsilon$$

The above Theorem indicates that the backpropagation algorithm is locally $H^\infty$ optimal. In other words for $w_{-1}$ sufficiently close to $w$, and for sufficiently small disturbance, the ratio in (6) can be made arbitrarily close to one. Note that the conditions on $w$ and $v_i$ are reasonable, since if for example $w$ is too far from $w_{-1}$, or if some $v_i$ is too large, then it is well known that backpropagation may get stuck in a local minimum, in which case the ratio in (6) may get arbitrarily large.

As before (11) gives an upper bound on the learning rate $\mu$, and indicates why backpropagation behaves poorly if the learning rate is too large.

If there is no disturbance in (5) we have the following

**Corollary 2** *If in addition to the assumptions in Theorem 3 there is no disturbance in (5), then for every $\epsilon > 0$ there exists a $\delta > 0$ such that for all $|w - w_{-1}| < \delta$, the backpropagation algorithm will yield $\| e' \|_2^2 \leq \mu^{-1}\delta(1 + \epsilon)$, meaning that the prediction error converges to zero. Moreover $w_i$ will converge to $w$.*

Again provided (11) is satisfied, the larger $\mu$ is the faster the convergence will be.

## 4  Discussion and Conclusion

The results presented in this paper give some new insights into the behaviour of instantaneous gradient-based adaptive algorithms. We showed that if the underlying observation model is linear then LMS is an $H^\infty$ optimal estimator, whereas if the underlying observation model is nonlinear then the backpropagation algorithm is locally $H^\infty$ optimal. The $H^\infty$ optimality of these algorithms explains their inherent robustness to unknown disturbances and modelling errors, as opposed to other estimation algorithms for which such bounds are not guaranteed.

Note that if one considers the transfer operator from the disturbances to the prediction errors, then LMS (backpropagation) is $H^\infty$ optimal (locally), over all *causal* estimators. This indicates that our result is most applicable in situations where one is confronted with real-time data and there is no possiblity of storing the training patterns. Such cases arise when one uses adaptive filters or adaptive neural networks for adaptive noise cancellation, channel equalization, real-time control, and undoubtedly many other situations. This is as opposed to pattern recognition, where one has a set of training patterns and repeatedly retrains the network until a desired performance is reached.

Moreover, we also showed that the $H^\infty$ optimality result leads to convergence proofs for the LMS and backpropagation algorithms in the absence of disturbances. We can pursue this line of thought further and argue why choosing large learning rates increases the resistance of backpropagation to local minima, but we shall not do so due to lack of space.

In conclusion these results give a new interpretation of the LMS and backpropagation algorithms, which we believe should be worthy of further scrutiny.

## Acknowledgements

This work was supported in part by the Air Force Office of Scientific Research, Air Force Systems Command under Contract AFOSR91-0060 and in part by a grant from Rockwell International Inc.

## References

J. A. Ball and J. W. Helton. (1992) Nonlinear $H^\infty$ control theory for stable plants. *Math. Control Signals Systems*, 5:233-261.

K. Glover and D. Mustafa. (1989) Derivation of the maximum entropy $H^\infty$ controller and a state space formula for its entropy. *Int. J. Control*, 50:899-916.

B. Hassibi, A. H. Sayed, and T. Kailath. (1993a) LMS is $H^\infty$ Optimal. *IEEE Conf. on Decision and Control*, 74-80, San Antonio, Texas.

B. Hassibi, A. H. Sayed, and T. Kailath. (1993b) Recursive linear estimation in Krein spaces - part II: Applications. *IEEE Conf. on Decision and Control*, 3495-3501, San Antonio, Texas.

S. Haykin. (1991) *Adaptive Filter Theory*. Prentice Hall, Englewood Cliffs, NJ.

D. E. Rumelhart, J. L. McClelland and the PDP Research Group. (1986) *Parallel distributed processing : explorations in the microstructure of cognition*. Cambridge, Mass. : MIT Press.

P. Whittle. (1990) *Risk Sensitive Optimal Control*. John Wiley and Sons, New York.

B. Widrow and M. E. Hoff, Jr. (1960) Adaptive switching circuits. *IRE WESCON Conv. Rec.*, Pt.4:96-104.

G. Zames. (1981) Feedback optimal sensitivity: model preference transformation, multiplicative seminorms and approximate inverses. *IEEE Trans. on Automatic Control*, AC-26:301-320.